# Algorithms and hardness results
# for parallel large margin learning

**Philip M. Long**
Google
plong@google.com

**Rocco A. Servedio**
Columbia University
rocco@cs.columbia.edu

## Abstract

We study the fundamental problem of learning an unknown large-margin half-space in the context of parallel computation.

Our main positive result is a parallel algorithm for learning a large-margin half-space that is based on interior point methods from convex optimization and fast parallel algorithms for matrix computations. We show that this algorithm learns an unknown $\gamma$-margin halfspace over $n$ dimensions using $\mathrm{poly}(n, 1/\gamma)$ processors and runs in time $\tilde{O}(1/\gamma) + O(\log n)$. In contrast, naive parallel algorithms that learn a $\gamma$-margin halfspace in time that depends polylogarithmically on $n$ have $\Omega(1/\gamma^2)$ runtime dependence on $\gamma$.

Our main negative result deals with boosting, which is a standard approach to learning large-margin halfspaces. We give an information-theoretic proof that in the original PAC framework, in which a weak learning algorithm is provided as an oracle that is called by the booster, boosting cannot be parallelized: the ability to call the weak learner multiple times in parallel within a single boosting stage does not reduce the overall number of successive stages of boosting that are required.

## 1 Introduction

In this paper we consider large-margin halfspace learning in the PAC model: there is a target halfspace $f(\mathbf{x}) = \mathrm{sign}(\mathbf{w} \cdot \mathbf{x})$, where $\mathbf{w}$ is an unknown unit vector, and an unknown probability distribution $\mathcal{D}$ over the unit ball $\mathbf{B}_n = \{\mathbf{x} \in \mathbf{R}^n : \|\mathbf{x}\|_2 \le 1\}$ which has support on $\{\mathbf{x} \in \mathbf{B}_n : |\mathbf{w} \cdot \mathbf{x}| \ge \gamma\}$. (Throughout this paper we refer to such a combination of target halfspace $f$ and distribution $\mathcal{D}$ as a $\gamma$-*margin halfspace*.) The learning algorithm is given access to labeled examples $(\mathbf{x}, f(\mathbf{x}))$ where each $\mathbf{x}$ is independently drawn from $\mathcal{D}$, and it must with high probability output a hypothesis $h : \mathbf{R}^n \to \{-1, 1\}$ that satisfies $\mathrm{Pr}_{\mathbf{x} \sim \mathcal{D}}[h(\mathbf{x}) \ne f(\mathbf{x})] \le \varepsilon$. Learning a large-margin halfspace is a fundamental problem in machine learning; indeed, one of the most famous algorithms in machine learning is the Perceptron algorithm [25] for this problem. PAC algorithms based on the Perceptron [17] run in $\mathrm{poly}(n, \frac{1}{\gamma}, \frac{1}{\varepsilon})$ time, use $O(\frac{1}{\varepsilon\gamma^2})$ labeled examples in $\mathbf{R}^n$, and learn an unknown $n$-dimensional $\gamma$-margin halfspace to accuracy $1 - \varepsilon$.

**A motivating question: achieving Perceptron's performance in parallel?** The last few years have witnessed a resurgence of interest in highly efficient parallel algorithms for a wide range of computational problems in many areas including machine learning [33, 32]. So a natural goal is to develop an efficient parallel algorithm for learning $\gamma$-margin halfspaces that matches the performance of the Perceptron algorithm. A well-established theoretical notion of efficient parallel computation is that an efficient parallel algorithm for a problem with input size $N$ is one that uses $\mathrm{poly}(N)$ processors and runs in parallel time $\mathrm{polylog}(N)$, see e.g. [12]. Since the input to the Perceptron algorithm is a sample of $\mathrm{poly}(\frac{1}{\varepsilon}, \frac{1}{\gamma})$ labeled examples in $\mathbf{R}^n$, we naturally arrive at the following:

| Algorithm | Number of processors | Running time |
|:---:|:---:|:---:|
| naive parallelization of Perceptron | $\text{poly}(n, 1/\gamma)$ | $\tilde{O}(1/\gamma^2) + O(\log n)$ |
| naive parallelization of [27] | $\text{poly}(n, 1/\gamma)$ | $\tilde{O}(1/\gamma^2) + O(\log n)$ |
| polynomial-time linear programming [2] | 1 | $\text{poly}(n, \log(1/\gamma))$ |
| This paper | $\text{poly}(n, 1/\gamma)$ | $\tilde{O}(1/\gamma) + O(\log n)$ |

Table 1: Bounds on various parallel algorithms for learning a $\gamma$-margin halfspace over $\mathbf{R}^n$.

> **Main Question:** Is there a learning algorithm that uses $\text{poly}(n, \frac{1}{\gamma}, \frac{1}{\varepsilon})$ processors and runs in time $\text{poly}(\log n, \log \frac{1}{\gamma}, \log \frac{1}{\varepsilon})$ to learn an unknown $n$-dimensional $\gamma$-margin halfspace to accuracy $1 - \varepsilon$?

(See [31] for a detailed definition of parallel learning algorithms; here we only recall that an efficient parallel learning algorithm's hypothesis must be efficiently evaluatable in parallel.) As Freund [10] has largely settled how the resources required by parallel algorithms scale with the accuracy parameter $\epsilon$ (see Lemma 6 below), our focus in this paper is on $\gamma$ and $n$, leading to the following:

> **Main Question (simplified):** Is there a learning algorithm that uses $\text{poly}(n, \frac{1}{\gamma})$ processors and runs in time $\text{poly}(\log n, \log \frac{1}{\gamma})$ to learn an unknown $n$-dimensional $\gamma$-margin halfspace to accuracy $9/10$?

This question, which we view as a fundamental open problem, inspired the research reported here.

**Prior results.** Table 1 summarizes the running time and number of processors used by various parallel algorithms to learn a $\gamma$-margin halfspace over $\mathbf{R}^n$. The naive parallelization of Perceptron in the first line of the table is an algorithm that runs for $O(1/\gamma^2)$ stages; in each stage it processes all of the $O(1/\gamma^2)$ examples simultaneously in parallel, identifies one that causes the Perceptron algorithm to update its hypothesis vector, and performs this update. We do not see how to obtain parallel time bounds better than $O(1/\gamma^2)$ from recent analyses of other algorithms based of gradient descent (such as [7, 8, 4]), some of which use assumptions incomparable in strength to the $\gamma$-margin condition studied here. The second line of the table corresponds to a similar naive parallelization of the boosting-based algorithm of [27] that achieves Perceptron-like performance for learning a $\gamma$-margin halfspace. It boosts for $O(1/\gamma^2)$ stages over a $O(1/\gamma^2)$-size sample; using one processor for each coordinate of each example, the running time bound is $\tilde{O}(1/\gamma^2) \cdot \log n$, using $\text{poly}(n, 1/\gamma)$ processors. (For both this algorithm and the Perceptron the time bound can be improved to $\tilde{O}(1/\gamma^2) + O(\log n)$ as claimed in the table by using an initial random projection step; we explain how to do this in Section 2.) The third line of the table, included for comparison, is simply a standard sequential algorithm for learning a halfspace based on polynomial-time linear programming executed on one processor, see e.g. [2, 14].

Efficient parallel algorithms have been developed for some simpler PAC learning problems such as learning conjunctions, disjunctions, and symmetric Boolean functions [31]. [6] gave efficient parallel PAC learning algorithms for some geometric constant-dimensional concept classes.

In terms of negative results for parallel learning, [31] shows that (under a complexity-theoretic assumption) there is no parallel algorithm using $\text{poly}(n)$ processors and $\text{polylog}(n)$ time that constructs a halfspace hypothesis that is consistent with a given linearly separable data set of $n$-dimensional labeled examples. This does not give a negative answer to the Main Question for several reasons: the Main Question allows any hypothesis representation (that can be efficiently evaluated in parallel), allows the number of processors to grow inverse polynomially with the margin parameter $\gamma$, and allows the final hypothesis to err on up to (say) 5% of the points in the data set.

**Our results.** Our main positive result is a parallel algorithm that uses $\text{poly}(n, 1/\gamma)$ processors to learn $\gamma$-margin halfspaces in parallel time $\tilde{O}(1/\gamma) + O(\log n)$ (see Table 1). We believe ours is the first algorithm that runs in time polylogarithmic in $n$ and subquadratic in $1/\gamma$. Our analysis can be modified to establish similar positive results for other formulations of the large-margin learning problem, including ones (see [28]) that have been tied closely to weak learnability (these modifications are not presented due to space constraints). In contrast, our main negative result is an

information-theoretic argument that suggests that such positive parallel learning results cannot be obtained by boosting alone. We show that if the weak learner must be called as an oracle, boosting cannot be parallelized: any parallel booster must perform $\Omega(1/\gamma^2)$ sequential stages of boosting a "black-box" $\gamma$-advantage weak learner in the worst case. This extends an earlier lower bound of Freund [10] for standard (sequential) boosters that can only call the weak learner once per stage.

## 2    A parallel algorithm for learning $\gamma$-margin halfspaces over $\mathbf{B}_n$

Our parallel algorithm is an amalgamation of existing tools from high-dimensional geometry, convex optimization, parallel algorithms for linear algebra, and learning theory. Roughly speaking the algorithm works as follows: given a data set of $m = \tilde{O}(1/\gamma^2)$ labeled examples from $\mathbf{B}_n \times \{-1, 1\}$, it begins by randomly projecting the examples down to $d = \tilde{O}(1/\gamma^2)$ dimensions. This essentially preserves the geometry so the resulting $d$-dimensional labeled examples are still linearly separable with margin $\Theta(\gamma)$. The algorithm then uses a variant of a linear programming algorithm of Renegar [24, 21] which, roughly speaking, solves linear programs with $m$ constraints to high accuracy using (essentially) $\sqrt{m}$ stages of Newton's method. Within Renegar's algorithm we employ fast parallel algorithms for linear algebra [22] to carry out each stage of Newton's method in $\mathrm{polylog}(1/\gamma)$ parallel time steps. This suffices to learn the unknown halfspace to high constant accuracy (say 9/10); to get a $1 - \varepsilon$-accurate hypothesis we combine the above procedure with Freund's approach [10] for boosting accuracy that was mentioned in the introduction. The above sketch omits many details, including crucial issues of precision in solving the linear programs to adequate accuracy. In the rest of this section we address the necessary details in full and prove the following theorem:

**Theorem 1** *There is a parallel algorithm with the following performance guarantee: Let $f, \mathcal{D}$ define an unknown $\gamma$-margin halfspace over $\mathbf{B}_n$ as described in the introduction. The algorithm is given as input $\epsilon, \delta > 0$ and access to labeled examples $(\mathbf{x}, f(\mathbf{x}))$ that are drawn independently from $\mathcal{D}$. It runs in $O(((1/\gamma)\mathrm{polylog}(1/\gamma) + \log(n))\log(1/\epsilon) + \log\log(1/\delta))$ time, uses $\mathrm{poly}(n, 1/\gamma, 1/\epsilon, \log(1/\delta))$ processors, and with probability $1 - \delta$ it outputs a hypothesis $h$ satisfying $\Pr_{\mathbf{x} \sim \mathcal{D}}[h(\mathbf{x}) \neq f(\mathbf{x})] \leq \varepsilon$.*

We assume that the value of $\gamma$ is "known" to the algorithm, since otherwise the algorithm can use a standard "guess and check" approach trying $\gamma = 1, 1/2, 1/4$, etc., until it finds a value that works.

We first describe the tools from the literature that are used in the algorithm.

**Random projection.** We say that a *random projection matrix* is a matrix $A$ chosen uniformly from $\{-1, 1\}^{n \times d}$. Given such an $A$ and a unit vector $\mathbf{w} \in \mathbf{R}^n$ (recall that the target halfspace $f$ is $f(\mathbf{x}) = \mathrm{sign}(\mathbf{w} \cdot \mathbf{x})$), let $\mathbf{w}'$ denote $(1/\sqrt{d})\mathbf{w}A$. After transformation by $A$ the distribution $\mathcal{D}$ over $\mathbf{B}_n$ is transformed to a distribution $\mathcal{D}'$ over $\mathbf{R}^d$ in the natural way: a draw $\mathbf{x}'$ from $\mathcal{D}'$ is obtained by making a draw $\mathbf{x}$ from $\mathcal{D}$ and setting $\mathbf{x}' = (1/\sqrt{d})\mathbf{x}A$. We will use the following lemma from [1]:

**Lemma 1** *[1] Let $f(\mathbf{x}) = \mathrm{sign}(\mathbf{w} \cdot \mathbf{x})$ and $\mathcal{D}$ define a $\gamma$-margin halfspace as described in the introduction. For $d = O((1/\gamma^2)\log(1/\gamma))$, a random $n \times d$ projection matrix $A$ will with probability $99/100$ induce $\mathcal{D}'$ and $\mathbf{w}'$ as described above such that $\Pr_{\mathbf{x}' \sim \mathcal{D}'}\left[\left|\frac{\mathbf{w}'}{\|\mathbf{w}'\|} \cdot \mathbf{x}'\right| < \gamma/2 \ \ or \ \ \|\mathbf{x}'\|_2 > 2\right] \leq \gamma^4.$*

**Convex optimization.** We recall some tools we will use from convex optimization over $\mathbf{R}^d$ [24, 3].

Let $F$ be the convex barrier function $F(\mathbf{u}) = \sum_{i=1}^{d} \log\left(\frac{1}{(u_i - a_i)(b_i - u_i)}\right)$ (we specify the values $a_i < b_i$ below). Let $g(\mathbf{u})$ be the gradient of $F$ at $\mathbf{u}$; note that $g(\mathbf{u})_i = \frac{1}{b_i - u_i} - \frac{1}{u_i - a_i}$. Let $H(\mathbf{u})$ be the Hessian of $F$ at $\mathbf{u}$, let $\|\mathbf{v}\|_{\mathbf{u}} = \sqrt{\mathbf{v}^T H(\mathbf{u})\mathbf{v}}$, and let $n(\mathbf{u}) = -H(\mathbf{u})^{-1}g(\mathbf{u})$ be the Newton step at $\mathbf{u}$. For a linear subspace $L$ of $\mathbf{R}^d$, let $F_{|_L}$ be the restriction of $F$ to $L$, i.e. the function that evaluates to $F$ on $L$ and $\infty$ everywhere else.

We will apply interior point methods to approximately solve problems of the following form, where $a_1, ..., a_d, b_1, ..., b_d \in [-2, 2]$, $|b_i - a_i| \geq 2$ for all $i$, and $L$ is a subspace of $\mathbf{R}^d$:

$$\text{minimize } -u_1 \text{ such that } \mathbf{u} \in L \text{ and } a_i \leq x_i \leq b_i \text{ for all } i. \tag{1}$$

Let $\mathbf{z} \in \mathbf{R}^d$ be the minimizer, and let $\mathsf{opt}$ be the optimal value of (1).

The algorithm we analyze minimizes $F_\eta(\mathbf{u}) \stackrel{\text{def}}{=} -\eta u_1 + F_{|_L}(\mathbf{u})$ for successively larger values of $\eta$. Let $\mathbf{z}(\eta)$ be the minimizer of $F_\eta$, let $\mathsf{opt}_\eta = F_\eta(\mathbf{z}(\eta))$, and let $n_\eta(\mathbf{u})$ be its Newton step. (To keep the notation clean, the dependence on $L$ is suppressed from the notation.)

As in [23], we periodically round intermediate solutions to keep the bit complexity under control. The analysis of such rounding in [23] requires a problem transformation which does not preserve the large-margin condition that we need for our analysis, so we give a new analysis, using tools from [24], and a simpler algorithm. It is easier to analyze the effect of the rounding on the quality of the solution than on the progress measure used in [24]. Fortunately, [3] describes an algorithm that can go from an approximately optimal solution to a solution with a good measure of progress while controlling the bit complexity of the output. The algorithm repeatedly finds the direction of the Newton step, and then performs a line search to find the approximately optimal step size.

**Lemma 2 ([3, Section 9.6.4])** *There is an algorithm $A_{bt}$ with the following property. Suppose for any $\eta > 0$, $A_{bt}$ is given $\mathbf{u}$ with rational components such that $F_\eta(\mathbf{u}) - \mathsf{opt}_\eta \leq 2$. Then after constantly many iterations of Newton's method and back-tracking line search, $A_{bt}$ returns an $\mathbf{u}^+$ that (i) satisfies $||n_\eta(\mathbf{u}^+)||_{\mathbf{u}^+} \leq 1/9$; and (ii) has rational components that have bit-length bounded by a polynomial in d, the bit length of $\mathbf{u}$, and the bit length of the matrix $A$ such that $L = \{\mathbf{v} : A\mathbf{v} = 0\}$.*[1]

We analyze the following variant of the usual central path algorithm for linear programming, which we call $A_{\mathrm{cpr}}$. It takes as input a precision parameter $\alpha$ and outputs the final $\mathbf{u}^{(k)}$.

- Set $\eta_1 = 1$, $\beta = 1 + \frac{1}{8\sqrt{2d}}$ and $\epsilon = \frac{1}{\lceil 2\sqrt{d}(5d/\alpha + \sqrt{d}2^{10d/\alpha + 2d + 1})\rceil}$.

- Given $\mathbf{u}$ as input, run $A_{bt}$ starting with $\mathbf{u}$ to obtain $\mathbf{u}^{(1)}$ such that $||n_{\eta_1}(\mathbf{u}^{(1)})||_{\mathbf{u}^{(1)}} \leq 1/9$.

- For $k$ from 2 to $1 + \lceil \frac{\log(4d/\alpha)}{\log(\beta)} \rceil$ perform the following steps (i)–(iv): (i) set $\eta_k = \beta\eta_{k-1}$; (ii) set $\mathbf{w}^{(k)} = \mathbf{u}^{(k-1)} + n_{\eta_k}(\mathbf{u}^{(k-1)})$ (i.e. do one step of Newton's method); (iii) form $\mathbf{r}^{(k)}$ by rounding each component of $\mathbf{w}^{(k)}$ to the nearest multiple of $\epsilon$, and then projecting back onto $L$; (iv) Run $A_{bt}$ starting with $\mathbf{r}^{(k)}$ to obtain $\mathbf{u}^{(k)}$ such that $||n_{\eta_k}(\mathbf{u}^{(k)})||_{\mathbf{u}^{(k)}} \leq 1/9$.

The following lemma, implicit[2] in [3, 24], bounds the quality of the solutions in terms of the progress measure $||n_{\eta_k}(\mathbf{u})||_{\mathbf{u}}$.

**Lemma 3** *If $\mathbf{u} \in L$ and $||n_{\eta_k}(\mathbf{u})||_{\mathbf{u}} \leq 1/9$, then $F_{\eta_k}(\mathbf{u}) - \mathsf{opt}_{\eta_k} \leq ||n_{\eta_k}(\mathbf{u})||_{\mathbf{u}}^2$ and $-u_1 - \mathsf{opt} \leq \frac{4d}{\eta_k}$.*

The following key lemma shows that rounding intermediate solutions does not do too much harm:

**Lemma 4** *For any k, if $F_{\eta_k}(\mathbf{w}^{(k)}) \leq \mathsf{opt}_{\eta_k} + 1/9$, then $F_{\eta_k}(\mathbf{r}^{(k)}) \leq \mathsf{opt}_{\eta_k} + 1$.*

**Proof:** Fix $k$, and note that $\eta_k = \beta^{k-1} \leq 5d/\alpha$. We henceforth drop $k$ from all notation.

First, we claim that
$$\kappa = \min_i\{|a_i - w_i|, |b_i - w_i|\} \geq 2^{-2\eta - 2d - 1/9}. \tag{2}$$

Let $\mathbf{m} = ((a_1 + b_1)/2, ..., (a_d + b_d)/2)$. Since $F_\eta(\mathbf{w}) \leq \mathsf{opt}_\eta + 1/9$, we have $F_\eta(\mathbf{w}) \leq F_\eta(\mathbf{m}) + 1/9 \leq \eta + 1/9$. But minimizing each term of $F_\eta$ separately, we get $F_\eta(\mathbf{w}) \geq \log\left(\frac{1}{\kappa}\right) - 2d - \eta$. Combining this with the previous inequality and solving for $\kappa$ yields (2).

Since $||\mathbf{w} - \mathbf{r}|| \leq \epsilon\sqrt{d}$, recalling that $\epsilon \leq \frac{1}{2\sqrt{d}(5d/\alpha + \sqrt{d}2^{10d/\alpha + 2d + 1})}$, we have
$$\min_i\{|a_i - r_i|, |b_i - r_i|\} \geq 2^{-2\eta - 2d - 1/9} - \epsilon\sqrt{d} \geq 2^{-2\eta - 2d - 1}. \tag{3}$$

Now, define $\psi : \mathbf{R} \to \mathbf{R}$ by $\psi(t) = F_\eta \left( \mathbf{w} + t \frac{\mathbf{r} - \mathbf{w}}{||\mathbf{r} - \mathbf{w}||} \right)$. We have

$$F_\eta(\mathbf{r}) - F_\eta(\mathbf{w}) = \psi(||\mathbf{r} - \mathbf{w}||) - \psi(0) = \int_0^{||\mathbf{r} - \mathbf{w}||} \psi'(t)dt \leq ||\mathbf{r} - \mathbf{w}|| \max_t |\psi'(t)|. \quad (4)$$

Let $S$ be the line segment between $\mathbf{w}$ and $\mathbf{r}$. Since for each $t \in [0, ||\mathbf{r} - \mathbf{w}||]$ the value $\psi'(t)$ is a directional derivative of $F_\eta$ at some point of $S$, (4) implies that, for the gradient $g_\eta$ of $F_\eta$,

$$F_\eta(\mathbf{r}) - F_\eta(\mathbf{w}) \leq ||\mathbf{w} - \mathbf{r}|| \max\{||g_\eta(\mathbf{s})|| : \mathbf{s} \in S\}. \quad (5)$$

However (3) and (2) imply that $\min\{|a_i - s_i|, |b_i - s_i|\} \geq 2^{-2\eta - 2d - 1}$ for all $\mathbf{s} \in S$. Recalling that $g(\mathbf{u})_i = \frac{1}{b_i - u_i} - \frac{1}{u_i - a_i}$, this means that $||g_\eta(\mathbf{s})|| \leq \eta + \sqrt{d}2^{2\eta + 2d + 1}$ so that applying (5) we get $F_\eta(\mathbf{r}) - F_\eta(\mathbf{w}) \leq ||\mathbf{w} - \mathbf{r}||(\eta + \sqrt{d}2^{2\eta + 2d + 1})$. Since $||\mathbf{w} - \mathbf{r}|| \leq \epsilon\sqrt{d}$, we have $F_\eta(\mathbf{r}) - F_\eta(\mathbf{w}) \leq \epsilon\sqrt{d}(\eta + \sqrt{d}2^{2\eta + 2d + 1}) \leq \epsilon\sqrt{d}(5d/\alpha + \sqrt{d}2^{10d/\alpha + 2d + 1}) \leq 1/2$, and the lemma follows. ∎

**Fast parallel linear algebra: inverting matrices.** We will use an algorithm due to Reif [22]:

**Lemma 5 ([22])** *There is a $\mathrm{polylog}(d, L)$-time, $\mathrm{poly}(d, L)$-processor parallel algorithm which, given as input a $d \times d$ matrix $A$ with rational entries of total bit-length $L$, outputs $A^{-1}$.*

**Learning theory: boosting accuracy.** The following is implicit in the analysis of Freund [10].

**Lemma 6 ([10])** *Let $\mathcal{D}$ be a distribution over (unlabeled) examples. Let $A$ be a parallel learning algorithm such that for all $\mathcal{D}'$ with $\mathrm{support}(\mathcal{D}') \subseteq \mathrm{support}(\mathcal{D})$, given draws $(x, f(x))$ from $\mathcal{D}'$, with probability $9/10$ $A$ outputs a hypothesis with accuracy $9/10$ (w.r.t. $\mathcal{D}'$) using $\mathcal{P}$ processors in $\mathcal{T}$ time. Then there is a parallel algorithm $B$ that with probability $1 - \delta$ constructs a $(1 - \varepsilon)$-accurate hypothesis (w.r.t. $\mathcal{D}$) in $O(\mathcal{T}\log(1/\epsilon) + \log\log(1/\delta))$ time using $\mathrm{poly}(\mathcal{P}, 1/\epsilon, \log(1/\delta))$ processors.*

## 2.1 Proof of Theorem 1

As described at the start of this section, due to Lemma 6, it suffices to prove the lemma in the case that $\epsilon = 1/10$ and $\delta = 1/10$. We assume w.l.o.g. that $\gamma = 1/\mathrm{integer}$.

The algorithm first selects an $n \times d$ random projection matrix $A$ where $d = O(\log(1/\gamma)/\gamma^2)$. This defines a transformation $\Phi_A : \mathbf{B}^n \to \mathbf{R}^d$ as follows: given $\mathbf{x} \in \mathbf{B}^n$, the vector $\Phi_A(\mathbf{x}) \in \mathbf{R}^d$ is obtained by (i) rounding each $\mathbf{x}_i$ to the nearest integer multiple of $1/(4\lceil\sqrt{n/\gamma}\rceil)$; then (ii) setting $\mathbf{x}' = (1/2\sqrt{d})\mathbf{x}A$; and finally (iii) rounding each $\mathbf{x}'_i$ to the nearest multiple of $1/(8\lceil d/\gamma\rceil)$. Given $\mathbf{x}$ it is easy to compute $\Phi_A(\mathbf{x})$ using $O(n\log(1/\gamma)/\gamma^2)$ processors in $O(\log(n/\gamma))$ time. Let $\mathcal{D}'$ denote the distribution over $\mathbf{R}^d$ obtained by applying $\Phi_A$ to $\mathcal{D}$. Across all coordinates $\mathcal{D}'$ is supported on rational numbers with the same $\mathrm{poly}(1/\gamma)$ common denominator. By Lemma 1, with probability $99/100$ over $A$, the target-distribution pair $(\mathbf{w}' = (1/\sqrt{d})\mathbf{w}A, \mathcal{D}')$ satisfies

$$\Pr_{\mathbf{x}' \sim \mathcal{D}'} \left[ |\mathbf{x}' \cdot (\mathbf{w}'/||\mathbf{w}'||)| < \gamma' \stackrel{\mathrm{def}}{=} \gamma/8 \quad \text{or} \quad ||\mathbf{x}'||_2 > 1 \right] \leq \gamma^4. \quad (6)$$

The algorithm next draws $m = c\log(1/\gamma)/\gamma^2$ labeled training examples $(\Phi_A(\mathbf{x}), f(\mathbf{x}))$ from $\mathcal{D}'$; this can be done in $O(\log(n/\gamma))$ time using $O(n) \cdot \mathrm{poly}(1/\gamma)$ processors as noted above. It then applies $A_{\mathrm{cpr}}$ to find a $d$-dimensional halfspace $h$ that classifies all $m$ examples correctly (more on this below). By (6), with probability at least (say) $29/30$ over the random draw of $(\Phi_A(\mathbf{x}_1), y_m), ..., (\Phi_A(\mathbf{x}_m), y_m)$, we have that $y_t(\mathbf{w}' \cdot \Phi_A(\mathbf{x}_t)) \geq \gamma$ and $||\Phi_A(\mathbf{x}_t)|| \leq 1$ for all $t = 1, \ldots, m$. Now the standard VC bound for halfspaces [30] applied to $h$ and $\mathcal{D}'$ implies that since $h$ classifies all $m$ examples correctly, with overall probability at least $9/10$ its accuracy is at least $9/10$ with respect to $\mathcal{D}'$, i.e. $\Pr_{\mathbf{x} \sim \mathcal{D}}[h(\Phi_A(\mathbf{x})) \neq f(\mathbf{x})] \leq 1/10$. So the hypothesis $h \circ \Phi_A$ has accuracy $9/10$ with respect to $\mathcal{D}$ with probability $9/10$ as required by Lemma 6.

It remains to justify the above claim about $A_{\mathrm{cpr}}$ classifying all examples correctly, and analyze the running time. More precisely we show that given $m = O(\log(1/\gamma)/\gamma^2)$ training examples in $\mathbf{B}_d$ with rational components that all have coordinates with a common denominator that is $\mathrm{poly}(1/\gamma)$ and are separable with a margin $\gamma' = \gamma/8$, $A_{\mathrm{cpr}}$ can be used to construct a $d$-dimensional halfspace that classifies them all correctly in $\tilde{O}(1/\gamma)$ parallel time using $\mathrm{poly}(1/\gamma)$ processors.

Given $(\mathbf{x}'_1, y_1), ..., (\mathbf{x}'_m, y_m) \in \mathbf{B}_d \times \{-1, 1\}$ satisfying the above conditions, we will apply algorithm $A_{\mathrm{cpr}}$ to the following linear program, called LP, with $\alpha = \gamma'/2$: "minimize $-s$ such that $y_t(\mathbf{v} \cdot \mathbf{x}'_t) - s_t = s$ and $0 \leq s_t \leq 2$ for all $t \in [m]$; $-1 \leq v_i \leq 1$ for all $i \in [d]$; and $-2 \leq s \leq 2$." Intuitively, $s$ is the minimum margin over all examples, and $s_t$ is the difference between each example's margin and $s$. The subspace $L$ is defined by the equality constraints $y_t(\mathbf{v} \cdot \mathbf{x}'_t) - s_t = s$.

Our analysis will conclude by applying the following lemma, with an initial solution of $s = -1$, $\mathbf{v} = \mathbf{0}$, and $s_t = 1$ for all $t$. (Note that $u_1$ corresponds to $s$.)

**Lemma 7** *Given any $d$-dimensional linear program in the form (1), and an initial solution $\mathbf{u} \in L$ such that $\min\{|u_i - a_i|, |u_i - b_i|\} \geq 1$ for all $i$, Algorithm $A_{\mathrm{cpr}}$ approximates the optimal solution to an additive $\pm\alpha$. It runs in $\sqrt{d} \cdot \mathrm{polylog}(d/\alpha)$ parallel time and uses $\mathrm{poly}(1/\alpha, d)$ processors.*

The LP constraints enforce that all examples are classified correctly with a margin of at least $s$. The feasible solution in which $\mathbf{v}$ is $\mathbf{w}'/||\mathbf{w}'||$, $s$ equals $\gamma'$ and $s_t = y_t(\mathbf{v} \cdot \mathbf{x}'_t) - s$ shows that the optimum solution of LP has value at most $-\gamma'$. So approximating the optimum to an additive $\pm\alpha = \pm\gamma'/2$ ensures that all examples are classified correctly, and it is enough to prove Lemma 7.

**Proof of Lemma 7:** First, we claim that, for all $k$, $||n_{\eta_k}(\mathbf{u}^{(k)})||_{\mathbf{u}^{(k)}} \leq 1/9$; given this, since the final value of $\eta_k$ is at least $4d/\alpha$, Lemma 3 implies that the solution is $\alpha$-close to optimal. We induct on $k$. For $k = 1$, since initially $\min_i\{|u_i - a_i|, |u_i - b_i|\} \geq 1$, we have $F(\mathbf{u}) \leq 0$, and, since $\eta_1 = 1$ and $u_1 \geq -1$ we have $F_{\eta_1}(\mathbf{u}) \leq 1$ and $\mathrm{opt}_{\eta_1} \geq -1$. So we can apply Lemma 2 to get the base case. Now, for the induction step, suppose $||n_{\eta_k}(\mathbf{u}^{(k)})||_{\mathbf{u}^{(k)}} \leq 1/9$. It then follows[3] from [24, page 46] that $||n_{\eta_{k+1}}(\mathbf{w}^{(k+1)})||_{\mathbf{w}^{(k+1)}} \leq 1/9$. Next, Lemmas 3 and 4 imply that $F_{\eta_{k+1}}(\mathbf{r}^{(k+1)}) - \mathrm{opt}_{\eta_{k+1}} \leq 1$. Then Lemma 2 gives $||n_{\eta_{k+1}}(\mathbf{u}^{(k+1)})||_{\mathbf{u}^{(k+1)}} \leq 1/9$ as required.

Next, we claim that the bit-length of all intermediate solutions is at most $\mathrm{poly}(d, 1/\gamma)$. This holds for $\mathbf{r}^{(k)}$, and follows for $\mathbf{u}^{(k)}$ and $\mathbf{w}^{(k)}$ because each of them is obtained from some $\mathbf{r}^{(k)}$ by performing a constant number of operations each of which blows up the bit length at most polynomially (see Lemma 2). Since each intermediate solution has polynomial bit length, the matrix inverses can be computed in $\mathrm{polylog}(d, 1/\gamma)$ time using $\mathrm{poly}(d, 1/\gamma)$ processors, by Lemma 5. The time bound then follows from the fact that there are at most $O(\sqrt{d} \log(d/\alpha))$ iterations. ∎

## 3 Lower bound for parallel boosting in the oracle model

Boosting is a widely used method for learning large-margin halfspaces. In this section we consider the question of whether boosting algorithms can be efficiently parallelized. We work in the original PAC learning setting [29, 16, 26] in which a weak learning algorithm is provided as an oracle that is called by the boosting algorithm, which must simulate a distribution over labeled examples for the weak learner. Our main result for this setting is that boosting is inherently sequential; being able to to call the weak learner multiple times in parallel within a single boosting stage does not reduce the overall number of sequential boosting stages that are required. In fact we show this in a very strong sense, by proving that a boosting algorithm that runs *arbitrarily* many copies of the weak learner in parallel in each stage cannot save *even one* stage over a sequential booster that runs the weak learner just once in each stage. This lower bound is unconditional and information-theoretic.

Below we first define the parallel boosting framework and give some examples of parallel boosters. We then state and prove our lower bound on the number of stages required by parallel boosters. A consequence of our lower bound is that $\Omega(\log(1/\varepsilon)/\gamma^2)$ stages of parallel boosting are required in order to boost a $\gamma$-advantage weak learner to achieve classification accuracy $1 - \varepsilon$ no matter how many copies of the weak learner are used in parallel in each stage.

Our definition of weak learning is standard in PAC learning, except that for our discussion it suffices to consider a single target function $f : X \to \{-1, 1\}$ over a domain $X$.

**Definition 1** *A $\gamma$-advantage weak learner $L$ is an algorithm that is given access to a source of independent random labeled examples drawn from an (unknown and arbitrary) probability distribution*

$\mathcal{P}$ over labeled examples $\{(x, f(x))\}_{x \in X}$. $L$ must[4] return a weak hypothesis $h : X \to \{-1, 1\}$ that satisfies $\Pr_{(x,f(x)) \leftarrow \mathcal{P}}[h(x) = f(x)] \geq 1/2 + \gamma$. Such an $h$ is said to have advantage $\gamma$ w.r.t. $\mathcal{P}$.

We fix $\mathcal{P}$ to henceforth denote the initial distribution over labeled examples, i.e. $\mathcal{P}$ is a distribution over $\{(x, f(x))\}_{x \in X}$ where the marginal distribution $\mathcal{P}_X$ may be an arbitrary distribution over $X$.

Intuitively, a boosting algorithm runs the weak learner repeatedly on a sequence of carefully chosen distributions to obtain a sequence of weak hypotheses, and combines the weak hypotheses to obtain a final hypothesis that has high accuracy under $\mathcal{P}$. We give a precise definition below, but first we give some intuition to motivate our definition. In stage $t$ of a parallel booster the boosting algorithm may run the weak learner many times in parallel using different probability distributions. The probability weight of a labeled example $(x, f(x))$ under a distribution constructed at the $t$-th stage of boosting may depend on the values of all the weak hypotheses from previous stages and on the value of $f(x)$, but may not depend on any of the weak hypotheses generated by any of the calls to the weak learner in stage $t$. No other dependence on $x$ is allowed, since intuitively the only interface that the boosting algorithm should have with each data point is through its label and the values of the weak hypotheses from earlier stages. We further observe that since the distribution $\mathcal{P}$ is the only source of labeled examples, a booster should construct the distributions at each stage by somehow "filtering" examples $(x, f(x))$ drawn from $\mathcal{P}$ based only on the value of $f(x)$ and the values of the weak hypotheses from previous stages. We thus define a parallel booster as follows:

**Definition 2 (Parallel booster)** *A $T$-stage parallel boosting algorithm with $N$-fold parallelism is defined by $TN$ functions $\{\alpha_{t,k}\}_{t \in [T], k \in [N]}$ and a (randomized) Boolean function $h$, where $\alpha_{t,k} : \{-1, 1\}^{(t-1)N+1} \to [0, 1]$ and $h : \{-1, 1\}^{TN} \to \{-1, 1\}$. In the $t$-th stage of boosting the weak learner is run $N$ times in parallel. For each $k \in [N]$, the distribution $\mathcal{P}_{t,k}$ over labeled examples that is given to the $k$-th run of the weak learner is as follows: a draw from $\mathcal{P}_{t,k}$ is made by drawing $(x, f(x))$ from $\mathcal{P}$ and accepting $(x, f(x))$ as the output of the draw from $\mathcal{P}_{t,k}$ with probability $p_x = \alpha_{t,k}(h_{1,1}(x), \dots, h_{t-1,N}(x), f(x))$ (and rejecting it and trying again otherwise). In stage $t$, for each $k \in [N]$ the booster gives the weak learner access to $\mathcal{P}_{t,k}$ as defined above and the weak learner generates a hypothesis $h_{t,k}$ that has advantage at least $\gamma$ w.r.t. $\mathcal{P}_{t,k}$.*

*After $T$ stages, $TN$ weak hypotheses $\{h_{t,k}\}_{t \in [T], k \in [N]}$ have been obtained from the weak learner. The final hypothesis of the booster is $H(x) := h(h_{1,1}(x), \dots, h_{T,N}(x))$, and its accuracy is $\min_{h_{t,k}} \Pr_{(x,f(x)) \leftarrow \mathcal{P}}[H(x) = f(x)]$, where the min is taken over all sequences of $TN$ weak hypotheses subject to the condition that each $h_{t,k}$ has advantage at least $\gamma$ w.r.t. $\mathcal{P}_{t,k}$.*

The parameter $N$ above corresponds to the number of processors that the parallel booster is using; we get a *sequential* booster when $N = 1$. Many of the most common PAC-model boosters in the literature are sequential boosters, such as [26, 10, 9, 11, 27, 5] and others. In [10] Freund gave a boosting algorithm and showed that after $T$ stages of boosting, his algorithm generates a final hypothesis that is guaranteed to have error at most $\text{vote}(\gamma, T) \stackrel{\text{def}}{=} \sum_{j=0}^{\lfloor T/2 \rfloor} \binom{T}{j} \left(\frac{1}{2} + \gamma\right)^j \left(1/2 - \gamma\right)^{T-j}$ (see Theorem 2.1 of [10]). Freund also gave a matching lower bound by showing (see his Theorem 2.4) that any $T$-stage sequential booster must have error at least as large as $\text{vote}(\gamma, T)$, and so consequently any sequential booster that generates a $(1 - \varepsilon)$-accurate final hypothesis must run for $\Omega(\log(1/\varepsilon)/\gamma^2)$ stages. Our Theorem 2 below extends this lower bound to parallel boosters.

Several parallel boosting algorithms have been given in the literature, including branching program [20, 13, 18, 19] and decision tree [15] boosters. All of these boosters take $\Omega(\log(1/\varepsilon)/\gamma^2)$ stages to learn to accuracy $1 - \varepsilon$; our theorem below implies that *any* parallel booster must run for $\Omega(\log(1/\varepsilon)/\gamma^2)$ stages no matter how many parallel calls to the weak learner are made per stage.

**Theorem 2** *Let $B$ be any $T$-stage parallel boosting algorithm with $N$-fold parallelism. Then for any $0 < \gamma < 1/2$, when $B$ is used to boost a $\gamma$-advantage weak learner the resulting final hypothesis may have error as large as $\text{vote}(\gamma, T)$ (see the discussion after Definition 2).*

We emphasize that Theorem 2 holds for any $\gamma$ and any $N$ that may depend on $\gamma$ in an arbitrary way.

The theorem is proved as follows: fix any $0 < \gamma < 1/2$ and fix $B$ to be any $T$-stage parallel boosting algorithm. We will exhibit a target function $f$ and a distribution $\mathcal{P}$ over $\{(x, f(x))_{x \in X}$, and describe a strategy that a weak learner $W$ can use to generate weak hypotheses $h_{t,k}$ that each have advantage at least $\gamma$ with respect to the distributions $\mathcal{P}_{t,k}$. We show that with this weak learner $W$, the resulting final hypothesis $H$ that $B$ outputs will have accuracy at most $1 - \text{vote}(\gamma, T)$.

We begin by describing the desired $f$ and $\mathcal{P}$. The domain $X$ of $f$ is $X = Z \times \Omega$, where $Z = \{-1, 1\}^n$ and $\Omega$ is the set of all $\omega = (\omega_1, \omega_2, \dots)$ where each $\omega_i$ belongs to $\{-1, 1\}$. The target function $f$ is simply $f(z, \omega) = z$. The distribution $\mathcal{P} = (\mathcal{P}^X, \mathcal{P}^Y)$ over $\{(x, f(x))\}_{x \in X}$ is defined as follows. A draw from $\mathcal{P}$ is obtained by drawing $x = (z, \omega)$ from $\mathcal{P}^X$ and returning $(x, f(x))$. A draw of $x = (z, \omega)$ from $\mathcal{P}^X$ is obtained by first choosing a uniform random value in $\{-1, 1\}$ for $z$, and then choosing $\omega_i \in \{-1, 1\}$ to equal $z$ with probability $1/2 + \gamma$ independently for each $i$. Note that under $\mathcal{P}$, given the label $z = f(x)$ of a labeled example $(x, f(x))$, each coordinate $\omega_i$ of $x$ is correct in predicting the value of $f(x, z)$ with probability $1/2 + \gamma$ independently of all other $\omega_j$'s.

We next describe a way that a weak learner $W$ can generate a $\gamma$-advantage weak hypothesis each time it is invoked by $B$. Fix any $t \in [T]$ and any $k \in [N]$. When $W$ is invoked with $\mathcal{P}_{t,k}$ it replies as follows (recall that for $x \in X$ we have $x = (z, \omega)$ as described above): (i) if $\Pr_{(x,f(x)) \leftarrow \mathcal{P}_{t,k}}[\omega_t = f(x)] \geq 1/2 + \gamma$ then the weak hypothesis $h_{t,k}(x)$ is the function "$\omega_t$," i.e. the $(t+1)$-st coordinate of $x$. Otherwise, (ii) the weak hypothesis $h_{t,k}(x)$ is "$z$," i.e. the first coordinate of $x$. (Note that since $f(x) = z$ for all $x$, this weak hypothesis has zero error under any distribution.)

It is clear that each weak hypothesis $h_{t,k}$ generated as described above indeed has advantage at least $\gamma$ w.r.t. $\mathcal{P}_{t,k}$, so the above is a legitimate strategy for $W$. The following lemma will play a key role:

**Lemma 8** *If $W$ never uses option (ii) then* $\Pr_{(x,f(x)) \leftarrow \mathcal{P}}[H(x) \neq f(x)] \geq \text{vote}(\gamma, T)$.

**Proof:** If the weak learner never uses option (ii) then $H$ depends only on variables $\omega_1, \dots, \omega_T$ and hence is a (randomized) Boolean function over these variables. Recall that for $(x = (z, \omega), f(x) = z)$ drawn from $\mathcal{P}$, each coordinate $\omega_1, \dots, \omega_T$ independently equals $z$ with probability $1/2 + \gamma$. Hence the optimal (randomized) Boolean function $H$ over inputs $\omega_1, \dots, \omega_T$ that maximizes the accuracy $\Pr_{(x,f(x)) \leftarrow \mathcal{P}}[H(x) = f(x)]$ is the (deterministic) function $H(x) = \text{Maj}(\omega_1, \dots, \omega_T)$ that outputs the majority vote of its input bits. (This can be easily verified using Bayes' rule in the usual "Naive Bayes" calculation.) The error rate of this $H$ is precisely the probability that at most $\lfloor T/2 \rfloor$ "heads" are obtained in $T$ independent $(1/2 + \gamma)$-biased coin tosses, which equals $\text{vote}(\gamma, T)$. $\blacksquare$

Thus it suffices to prove the following lemma, which we prove by induction on $t$:

**Lemma 9** *$W$ never uses option (ii) (i.e.* $\Pr_{(x,f(x)) \leftarrow \mathcal{P}_{t,k}}[\omega_t = f(x)] \geq 1/2 + \gamma$ *always).*

**Proof: Base case** ($t = 1$). For any $k \in [N]$, since $t = 1$ there are no weak hypotheses from previous stages, so the value of $p_x$ is determined by the bit $f(x) = z$ (see Definition 2). Hence $\mathcal{P}_{1,k}$ is a convex combination of two distributions which we call $\mathcal{D}_1$ and $\mathcal{D}_{-1}$. For $b \in \{-1, 1\}$, a draw of $(x = (z, \omega); f(x) = z)$ from $\mathcal{D}_b$ is obtained by setting $z = b$ and independently setting each coordinate $\omega_i$ equal to $z$ with probability $1/2 + \gamma$. Thus in the convex combination $\mathcal{P}_{1,k}$ of $\mathcal{D}_1$ and $\mathcal{D}_{-1}$, we also have that $\omega_1$ equals $z$ (i.e. $f(x)$) with probability $1/2 + \gamma$. So the base case is done.

**Inductive step** ($t > 1$). Fix any $k \in [N]$. The inductive hypothesis and the weak learner's strategy together imply that for each labeled example $(x = (z, \omega), f(x) = z)$, since $h_{s,\ell}(x) = \omega_s$ for $s < t$, the rejection sampling parameter $p_x = \alpha_{t,k}(h_{1,1}(x), \dots, h_{t-1,N}(x), f(x))$ is determined by $\omega_1, \dots, \omega_{t-1}$ and $z$ and does not depend on $\omega_t, \omega_{t+1}, \dots$. Consequently the distribution $\mathcal{P}_{t,k}$ over labeled examples is some convex combination of $2^t$ distributions which we denote $\mathcal{D}_{\bar{b}}$, where $\bar{b}$ ranges over $\{-1, 1\}^t$ corresponding to conditioning on all possible values for $\omega_1, \dots, \omega_{t-1}, z$. For each $\bar{b} = (b_1, \dots, b_t) \in \{-1, 1\}^t$, a draw of $(x = (z, \omega); f(x) = z)$ from $\mathcal{D}_{\bar{b}}$ is obtained by setting $z = b_t$, setting $(\omega_1, \dots, \omega_{t-1}) = (b_1, \dots, b_{t-1})$, and independently setting each other coordinate $\omega_j$ ($j \geq t$) equal to $z$ with probability $1/2 + \gamma$. In particular, because $\omega_t$ is conditionally independent of $\omega_1, \dots, \omega_{t-1}$ given $z$, $\Pr(\omega_t = z | \omega_1 = b_1, \dots, \omega_{t-1} = b_{t-1}) = \Pr(\omega_t = z) = 1/2 + \gamma$. Thus in the convex combination $\mathcal{P}_{t,k}$ of the different $\mathcal{D}_{\bar{b}}$'s, we also have that $\omega_t$ equals $z$ (i.e. $f(x)$) with probability $1/2 + \gamma$. This concludes the proof of the lemma and the proof of Theorem 2. $\blacksquare$

## Footnotes

[1]We note for the reader's convenience that $\lambda(\mathbf{u})$ in [3] is the same as our $||n(\mathbf{u}^+)||_{\mathbf{u}^+}$. The analysis on pages 503-505 of [3] shows that a constant number of iterations suffice. Each step is a projection of $H(\mathbf{u})^{-1}g(\mathbf{u})$ onto $L$, which can be seen to have bit-length bounded by a polynomial in the bit-length of $\mathbf{u}$. Composing polynomials constantly many times yields a polynomial, which gives the claimed bit-length bound for $\mathbf{u}^+$.

[2]The first inequality is (9.50) from [3]. The last line of p. 46 of [24] proves that $||n_{\eta_k}(\mathbf{u})||_{\mathbf{u}} \leq 1/9$ implies $||\mathbf{u} - \mathbf{z}(\eta)||_{\mathbf{z}(\eta)} \leq 1/5$ from which the second inequality follows by (2.14) of [24], using the fact that $\vartheta = 2d$ (proved on page 35 of [24]).

[3] Noting that $\vartheta \leq 2d$ [24, page 35].

[4] The usual definition of a weak learner would allow $L$ to fail with probability $\delta$. This probability can be made exponentially small by running $L$ multiple times so for simplicity we assume there is no failure probability.

# References

[1] R. Arriaga and S. Vempala. An algorithmic theory of learning: Robust concepts and random projection. In *Proc. 40th FOCS*, pages 616–623, 1999.

[2] A. Blumer, A. Ehrenfeucht, D. Haussler, and M. Warmuth. Learnability and the Vapnik-Chervonenkis dimension. *Journal of the ACM*, 36(4):929–965, 1989.

[3] S. P. Boyd and L. Vandenberghe. *Convex Optimization*. Cambridge, 2004.

[4] J. Bradley, A. Kyrola, D. Bickson, and C. Guestrin. Parallel coordinate descent for l1-regularized loss minimization. *ICML*, 2011.

[5] Joseph K. Bradley and Robert E. Schapire. Filterboost: Regression and classification on large datasets. In *NIPS*, 2007.

[6] N. Bshouty, S. Goldman, and H.D. Mathias. Noise-tolerant parallel learning of geometric concepts. *Inf. and Comput.*, 147(1):89 – 110, 1998.

[7] Michael Collins, Robert E. Schapire, and Yoram Singer. Logistic regression, adaboost and bregman distances. *Machine Learning*, 48(1-3):253–285, 2002.

[8] O. Dekel, R. Gilad-Bachrach, O. Shamir, and L. Xiao. Optimal distributed online prediction. *ICML*, 2011.

[9] C. Domingo and O. Watanabe. MadaBoost: a modified version of AdaBoost. In *Proc. 13th COLT*, pages 180–189, 2000.

[10] Y. Freund. Boosting a weak learning algorithm by majority. *Inf. and Comput.*, 121(2):256–285, 1995.

[11] Y. Freund. An adaptive version of the boost-by-majority algorithm. *Mach. Learn.*, 43(3):293–318, 2001.

[12] R. Greenlaw, H.J. Hoover, and W.L. Ruzzo. *Limits to Parallel Computation: P-Completeness Theory*. Oxford University Press, New York, 1995.

[13] A. Kalai and R. Servedio. Boosting in the presence of noise. *Journal of Computer & System Sciences*, 71(3):266–290, 2005.

[14] N. Karmarkar. A new polynomial time algorithm for linear programming. *Combinat.*, 4:373–395, 1984.

[15] M. Kearns and Y. Mansour. On the boosting ability of top-down decision tree learning algorithms. In *Proceedings of the Twenty-Eighth Annual Symposium on Theory of Computing*, pages 459–468, 1996.

[16] M. Kearns and U. Vazirani. *An Introduction to Computational Learning Theory*. MIT Press, Cambridge, MA, 1994.

[17] N. Littlestone. From online to batch learning. In *Proc. 2nd COLT*, pages 269–284, 1989.

[18] P. Long and R. Servedio. Martingale boosting. In *Proc. 18th Annual COLT*, pages 79–94, 2005.

[19] P. Long and R. Servedio. Adaptive martingale boosting. In *Proc. 22nd NIPS*, pages 977–984, 2008.

[20] Y. Mansour and D. McAllester. Boosting using branching programs. *Journal of Computer & System Sciences*, 64(1):103–112, 2002.

[21] Y. Nesterov and A. Nemirovskii. *Interior Point Polynomial Methods in Convex Programming: Theory and Applications*. Society for Industrial and Applied Mathematics, Philadelphia, 1994.

[22] John H. Reif. O(log2 n) time efficient parallel factorization of dense, sparse separable, and banded matrices. *SPAA*, 1994.

[23] J. Renegar. A polynomial-time algorithm, based on Newton's method, for linear programming. *Mathematical Programming*, 40:59–93, 1988.

[24] James Renegar. *A mathematical view of interior-point methods in convex optimization*. Society for Industrial and Applied Mathematics, 2001.

[25] F. Rosenblatt. The Perceptron: a probabilistic model for information storage and organization in the brain. *Psychological Review*, 65:386–407, 1958.

[26] R. Schapire. The strength of weak learnability. *Machine Learning*, 5(2):197–227, 1990.

[27] R. Servedio. Smooth boosting and learning with malicious noise. *JMLR*, 4:633–648, 2003.

[28] S. Shalev-Shwartz and Y. Singer. On the equivalence of weak learnability and linear separability: New relaxations and efficient boosting algorithms. *Machine Learning*, 80(2):141–163, 2010.

[29] L. Valiant. A theory of the learnable. *Communications of the ACM*, 27(11):1134–1142, 1984.

[30] V. Vapnik. *Statistical Learning Theory*. Wiley-Interscience, New York, 1998.

[31] J. S. Vitter and J. Lin. Learning in parallel. *Inf. Comput.*, 96(2):179–202, 1992.

[32] DIMACS 2011 Workshop. Parallelism: A 2020 Vision. 2011.

[33] NIPS 2009 Workshop. Large-Scale Machine Learning: Parallelism and Massive Datasets. 2009.

